# Measuring Neural Synchrony by Message Passing

**Justin Dauwels**
Amari Research Unit
RIKEN Brain Science Institute
Wako-shi, Saitama, Japan
justin@dauwels.com

**François Vialatte, Tomasz Rutkowski, and Andrzej Cichocki**
Advanced Brain Signal Processing Laboratory
RIKEN Brain Science Institute
Wako-shi, Saitama, Japan
{fvialatte,tomek,cia}@brain.riken.jp

## Abstract

A novel approach to measure the interdependence of two time series is proposed, referred to as "stochastic event synchrony" (SES); it quantifies the alignment of two point processes by means of the following parameters: time delay, variance of the timing jitter, fraction of "spurious" events, and average similarity of events. SES may be applied to generic one-dimensional and multi-dimensional point processes, however, the paper mainly focusses on point processes in time-frequency domain. The average event similarity is in that case described by two parameters: the average frequency offset between events in the time-frequency plane, and the variance of the frequency offset ("frequency jitter"); SES then consists of five parameters in total. Those parameters quantify the synchrony of oscillatory events, and hence, they provide an alternative to existing synchrony measures that quantify amplitude or phase synchrony. The pairwise alignment of point processes is cast as a statistical inference problem, which is solved by applying the max-product algorithm on a graphical model. The SES parameters are determined from the resulting pairwise alignment by maximum a posteriori (MAP) estimation. The proposed interdependence measure is applied to the problem of detecting anomalies in EEG synchrony of Mild Cognitive Impairment (MCI) patients; the results indicate that SES significantly improves the sensitivity of EEG in detecting MCI.

## 1 Introduction

Synchrony is an important topic in neuroscience. For instance, it is hotly debated whether the synchronous firing of neurons plays a role in cognition [1] and even in consciousness [2]. The synchronous firing paradigm has also attracted substantial attention in both the experimental (e.g., [3]) and the theoretical neuroscience literature (e.g., [4]). Moreover, medical studies have reported that many neurophysiological diseases (such as Alzheimer's disease) are often associated with abnormalities in neural synchrony [5, 6].

In this paper, we propose a novel measure to quantify the interdependence between point processes, referred to as "stochastic event synchrony" (SES); it consists of the following parameters: time delay, variance of the timing jitter, fraction of "spurious" events, and average similarity of the events. The pairwise alignment of point processes is cast as a statistical inference problem, which is solved by applying the max-product algorithm on a graphical model [7]. In the case of one-dimensional point processes, the graphical model is cycle-free and statistical inference is exact, whereas for

multi-dimensional point processes, exact inference becomes intractable; the max-product algorithm is then applied on a cyclic graphical model, which not necessarily yields the optimal alignment [7]. Our experiments, however, indicate that the it finds reasonable alignments in practice. The SES parameters are determined from the resulting pairwise alignments by maximum a posteriori (MAP) estimation.

The proposed method may be helpful to detect mental disorders such as Alzheimer's disease, since mental disorders are often associated with abnormal blood and neural activity flows, and changes in the synchrony of brain activity (see, e.g., [5, 6]). In this paper, we will present promising results on the early prediction of Alzheimer's disease from EEG signals based on SES.

This paper is organized as follows. In the next section, we introduce SES for the case of one-dimensional point processes. In Section 3, we consider the extension to multi-dimensional point processes. In Section 4, we use our measure to detect abnormalities in the EEG synchrony of Alzheimer's disease patients.

## 2  One-Dimensional Point Processes

Let us consider the one-dimensional point processes ("event strings") $X$ and $X'$ in Fig. 1(a) (ignore $Y$ and $Z$ for now). We wish to quantify to which extent $X$ and $X'$ are synchronized. Intuitively speaking, two event strings can be considered as synchronous (or "locked") if they are identical apart from: (i) a time shift $\delta_t$; (ii) small deviations in the event occurrence times ("event timing jitter"); (iii) a few event insertions and/or deletions. More precisely, for two event strings to be synchronous, the event timing jitter should be significantly smaller than the average inter-event time, and the number of deletions and insertions should comprise only a small fraction of the total number of events. This intuitive concept of synchrony is illustrated in Fig. 1(a). The event string $X'$ is obtained from event string $X$ by successively shifting $X$ over $\delta_t$ (resulting in $Y$), slightly perturbing the event occurrence times (resulting in $Z$), and eventually, by adding (plus sign) and deleting (minus sign) events, resulting in $X'$. Adding and deleting events in $Z$ leads to "spurious" events in $X$ and $X'$ (see Fig. 1(a); spurious events are marked in red): a spurious event in $X$ is an event that cannot be paired with an event in $X'$ and vice versa.

The above intuitive reasoning leads to our novel measure for synchrony between two event strings, i.e., "stochastic event synchrony" (SES); for the one-dimensional case, it is defined as the triplet $(\delta_t, s_t, \rho_{\text{spur}})$, where $s_t$ is the variance of the (event) timing jitter, and $\rho_{\text{spur}}$ is the percentage of spurious events

$$\rho_{\text{spur}} \overset{\triangle}{=} \frac{n_{\text{spur}} + n'_{\text{spur}}}{n + n'}, \tag{1}$$

with $n$ and $n'$ the total number of events in $X$ and $X'$ respectively, and $n_{\text{spur}}$ and $n'_{\text{spur}}$ the total number of spurious events in $X$ and $X'$ respectively. SES is related to the metrics ("distances") proposed in [9]; those metrics are single numbers that quantify the synchrony between event strings. In contrast, we characterize synchrony by means of three parameters, which allows us to distinguish different types of synchrony (see [10]). We compute those three parameters by performing inference in a probabilistic model. In order to describe that model, we consider Fig. 1(b), which shows a symmetric procedure to generate $X$ and $X'$. First, one generates an event string $V$ of length $\ell$, where the events $V_k$ are mutually independent and uniformly distributed in $[0, T_0]$. The strings $Z$ and $Z'$ are generated by delaying $V$ over $-\delta_t/2$ and $\delta_t/2$ respectively and by (slightly) perturbing the resulting event occurrence times (variance of timing jitter equals $s_t/2$). The sequences $X$ and $X'$ are obtained from $Z$ and $Z'$ by removing some of the events; more precisely, from each pair $(Z_k, Z'_k)$, either $Z_k$ or $Z'_k$ is removed with probability $p_s$.

This procedure amounts to the statistical model:

$$p(x, x', b, b', v, \delta_t, s_t, \ell) = p(x|b, v, \delta_t, s_t)p(x'|b', v, \delta_t, s_t)p(b, b'|\ell)p(v|\ell)p(\ell)p(\delta_t)p(s_t), \tag{2}$$

where $b$ and $b'$ are binary strings that indicate whether the events in $X$ and $X'$ are spurious ($B_k = 1$ if $X_k$ is spurious, $B_k = 0$ otherwise; likewise for $B'_k$); the length $\ell$ has a geometric prior $p(\ell) = (1-\lambda)\lambda^\ell$ with $\lambda \in (0, 1)$, and $p(v|\ell) = T_0^{-\ell}$. The prior on the binary strings $b$ and $b'$ is given by

$$p(b, b'|\ell) = (1 - p_s)^{n+n'} p_s^{2\ell - n - n'} = (1 - p_s)^{n+n'} p_s^{n_{\text{spur}}^{\text{tot}}}, \tag{3}$$

with $n_{\text{spur}}^{\text{tot}} = n_{\text{spur}} + n'_{\text{spur}} = 2\ell - n - n'$ the total number of spurious events in $X$ and $X'$, $n_{\text{spur}} = \sum_{k=1}^{n} b_k = \ell - n'$ the number of spurious events in $X$, and likewise $n'_{\text{spur}}$, the number of spurious events in $X'$. The conditional distributions in $X$ and $X'$ are equal to:

$$p(x|b, v, \delta_t, s_t) = \prod_{k=1}^{n} \left( \mathcal{N}\left(x_k - v_{i_k}; -\frac{\delta_t}{2}, \frac{s_t}{2}\right) \right)^{1-b_k} \tag{4}$$

$$p(x'|b', v, \delta_t, s_t) = \prod_{k=1}^{n'} \left( \mathcal{N}\left(x'_k - v_{i'_k}; \frac{\delta_t}{2}, \frac{s_t}{2}\right) \right)^{1-b'_k}, \tag{5}$$

where $V_{i_k}$ is the event in $V$ that corresponds to $X_k$ (likewise $V_{i'_k}$), and $\mathcal{N}(x; m, s)$ is a univariate Gaussian distribution with mean $m$ and variance $s$. Since we do not wish/need to encode prior information about $\delta_t$ and $s_t$, we adopt improper priors $p(\delta_t) = 1 = p(s_t)$.

Eventually, marginalizing (2) w.r.t. $v$ results in the model:

$$p(x, x', b, b', \delta_t, s_t, \ell) = \int p(x, x', b, b', v, \delta_t, s_t, \ell) dv \propto \beta^{n_{\text{spur}}^{\text{tot}}} \prod_{k=1}^{n_{\text{non-spur}}} \mathcal{N}(x'_{j'_k} - x_{j_k}; \delta_t, s_t), \tag{6}$$

with $(x_{j_k}, x'_{j'_k})$ the pairs of non-spurious events, $n_{\text{non-spur}} = n + n' - \ell$ the total number of non-spurious event pairs, and $\beta = p_s \sqrt{\frac{\lambda}{T_0}}$; in the example of Fig. 1(b), $J = (1, 2, 3, 5, 6, 7, 8)$, $J' = (2, 3, 4, 5, 6, 7, 8)$, and $n_{\text{non-spur}} = 7$. In the following, we will denote model (6) by $p(x, x', j, j', \delta_t, s_t)$ instead of $p(x, x', b, b', \delta_t, s_t, \ell)$, since for given $x, x', b$, and $b'$ (and hence given $n, n'$, and $n_{\text{non-spur}}$), the length $\ell$ is fully determined, i.e., $\ell = n + n' - n_{\text{non-spur}}$; moreover, it is more natural to describe the model in terms of $J$ and $J'$ instead of $B$ and $B'$ (cf. RHS of (6)). Note that $B$ and $B'$ can directly be obtained from $J$ and $J'$.

It also noteworthy that $T_0$, $\lambda$ and $p_s$ do not need to be specified individually, since they appear in (6) only through $\beta$. The latter serves in practice as a knob to control the number of spurious events.

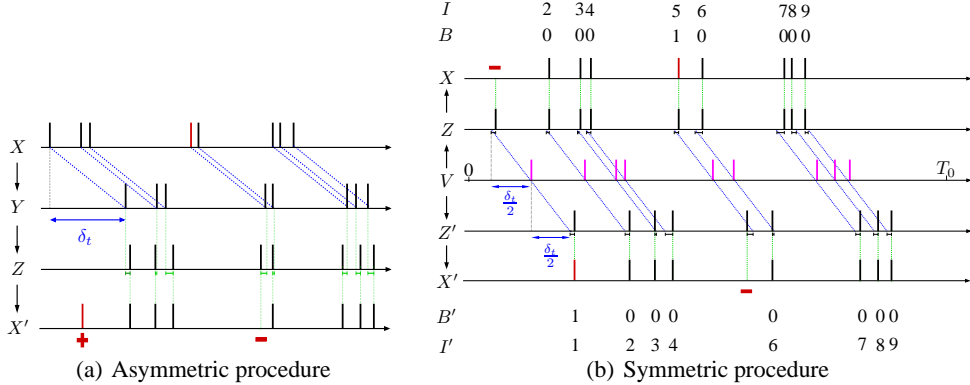

(a) Asymmetric procedure  (b) Symmetric procedure

Figure 1: One-dimensional stochastic event synchrony.

Given event strings $X$ and $X'$, we wish to determine the parameters $\delta_t$ and $s_t$, and the hidden variables $B$ and $B'$; the parameter $\rho_{\text{spur}}$ (cf. (1)) can obtained from the latter :

$$\rho_{\text{spur}} \triangleq \frac{\sum_{k=1}^{n} b_k + \sum_{k=1}^{n'} b'_k}{n + n'}. \tag{7}$$

There are various ways to solve this inference problem, but perhaps the most natural one is cyclic maximization: first one chooses initial values $\hat{\delta}_t^{(0)}$ and $\hat{s}_t^{(0)}$, then one alternates the following two update rules until convergence (or until the available time has elapsed):

$$(\hat{j}^{(i+1)}, \hat{j}'^{(i+1)}) = \underset{b, b'}{\operatorname{argmax}} \, p(x, x', j, j', \hat{\delta}_t^{(i)}, \hat{s}_t^{(i)}) \tag{8}$$

$$(\hat{\delta}_t^{(i+1)}, \hat{s}_t^{(i+1)}) = \underset{\delta_t, s_t}{\operatorname{argmax}} \, p(x, x', \hat{j}^{(i+1)}, \hat{j}'^{(i+1)}, \delta_t, s_t). \tag{9}$$

The update (9) is straightforward, it amounts to the empirical mean and variance, computed over the non-spurious events. The update (8) can readily be carried out by applying the Viterbi algorithm ("dynamic programming") on an appropriate trellis (with the pairs of non-spurious events $(x_{j_k}, x'_{j'_k})$ as states), or equivalently, by applying the max-product algorithm on a suitable factor graph [7]; the procedure is similar to dynamic time warping [8].

## 3 Multi-Dimensional Point Processes

In this section, we will focus on the interdependence of multi-dimensional point processes. As a concrete example, we will consider multi-dimensional point processes in time-frequency domain; the proposed algorithm, however, is not restricted to that particular situation, it is applicable to generic multi-dimensional point processes.

Suppose that we are given a pair of (continuous-time) signals, e.g., EEG signals recorded from two different channels. As a first step, the time-frequency ("wavelet") transform of each signal is approximated as a sum of (half-ellipsoid) basis functions, referred to as "bumps" (see Fig. 2 and [17]); each bump is described by five parameters: time $X$, frequency $F$, width $\Delta X$, height $\Delta F$, and amplitude $W$. The resulting bump models $Y = ((X_1, F_1, \Delta X_1, \Delta F_1, W_1), \ldots, (X_n, F_n, \Delta X_n, \Delta F_n, W_n))$ and $Y' = ((X'_1, F'_1, \Delta X'_1, \Delta F'_1, W'_1), \ldots, (X'_{n'}, F'_{n'}, \Delta X'_{n'}, \Delta F'_{n'}, W'_{n'}))$, representing the most prominent oscillatory activity, are thus 5-dimensional point processes. Our extension of stochastic event synchrony to multi-dimensional point processes (and bump models in particular) is derived from the following observation (see Fig. 3): bumps in one time-frequency map may not be present in the other map ("spurious" bumps); other bumps are present in both maps ("non-spurious bumps"), but appear at slightly different positions on the maps. The black lines in Fig. 3 connect the centers of non-spurious bumps, and hence, visualize the offset between pairs of non-spurious bumps. We quantify the interdependence between two bump models by five parameters, i.e., the parameters $\rho_{\text{spur}}$, $\delta_t$, and $s_t$ introduced in Section 2, in addition to:

- $\delta_f$: the average frequency offset between non-spurious bumps,
- $s_f$: the variance of the frequency offset between non-spurious bumps.

We determine the alignment of two bump models in addition to the 5 above parameters by an inference algorithm similar to the one of Section 2, as we will explain in the following; we will use the notation $\theta = (\delta_t, s_t, \delta_f, s_f)$. Model (6) may naturally be extended in time-frequency domain as:

$$p(y, y', j, j', \theta) \propto \beta^{n_{\text{spur}}^{\text{tot}}} \prod_{k=1}^{n_{\text{non-spur}}} \mathcal{N}\Big(\frac{x'_{k'} - x_k}{\Delta x_k + \Delta x'_{k'}}; \delta_t, s_t\Big) \mathcal{N}\Big(\frac{f'_{k'} - f_k}{\Delta f_k + \Delta f'_{k'}}; \delta_f, s_f\Big)$$
$$\cdot p(\delta_t)p(s_t)p(\delta_f)p(s_f), \tag{10}$$

where the offset $x'_{k'} - x_k$ in time and offset $f'_{k'} - f_k$ in frequency are normalized by the width and height respectively of the bumps; we will elaborate on the priors on the parameters $\theta$ later on. In principle, one may determine the sequences $J$ and $J'$ and the parameters $\theta$ by cyclic maximization along the lines of (8) and (9). In the multi-dimensional case, however, the update (8) is no longer tractable: one needs to allow permutations of events, the indices $j_k$ and $j'_{k'}$ are no longer necessarily monotonically increasing, and as a consequence, the state space becomes drastically larger. As a result, the Viterbi algorithm (or equivalently, the max-product algorithm applied on cycle-free factor graph of model (10)) becomes impractical.

We solve this problem by applying the max-product algorithm on a *cyclic* factor graph of the system at hand, which will amount to a suboptimal but practical procedure to obtain pairwise alignments of multi-dimensional point processes (and bump models in particular). To this end, we introduce a representation of model (10) that is naturally represented by a cyclic graph: for each pair of events $Y_k$ and $Y'_{k'}$, we introduce a binary variable $C_{kk'}$ that equals one if $Y_k$ and $Y'_{k'}$ form pair of non-spurious events and is zero otherwise. Since each event in $Y$ associated to at most one event in $Y'$, we have the constraints:

$$\sum_{k'=1}^{n'} C_{1k'} \triangleq S_1 \in \{0, 1\}, \sum_{k'=1}^{n'} C_{2k'} \triangleq S_2 \in \{0, 1\}, \ldots, \sum_{k'=1}^{n'} C_{nk'} \triangleq S_n \in \{0, 1\}, \tag{11}$$

and similarly, each event in $Y'$ is associated to at most one event in $Y$, which is expressed by a similar set of constraints. The sequences $S$ and $S'$ are related to the sequences $B$ and $B'$ (cf. Section 2): $B_k = 1 - S_k$ and $B_k' = 1 - S_k'$. In this representation, the global statistical model (10) can be cast as:

$$p(y, y', b, b', c, \theta) \propto \prod_{k=1}^{n} (\beta\delta[b_k - 1] + \delta[b_k]) \prod_{k'=1}^{n'} (\beta\delta[b_k' - 1] + \delta[b_k'])$$

$$\cdot \prod_{k=1}^{n} \prod_{k'=1}^{n'} \left( \mathcal{N}\left(\frac{x_{k'}' - x_k}{\Delta x_k + \Delta x_{k'}'}; \delta_t, s_t\right) \mathcal{N}\left(\frac{f_{k'}' - f_k}{\Delta f_k + \Delta f_{k'}'}; \delta_f, s_f\right) \right)^{c_{kk'}} p(\delta_t)p(s_t)p(\delta_f)p(s_f)$$

$$\cdot \prod_{k=1}^{n} \left(\delta[b_k + \sum_{k'=1}^{n'} c_{kk'} - 1]\right) \prod_{k'=1}^{n'} \left(\delta[b_{k'}' + \sum_{k=1}^{n} c_{kk'} - 1]\right). \tag{12}$$

Since we do not need to encode prior information about $\delta_t$ and $\delta_f$, we choose improper priors $p(\delta_t) = 1 = p(\delta_f)$. On the other hand, we have prior knowledge about $s_t$ and $s_f$. Indeed, we expect a bump in one time-frequency map to appear in the other map at about the same *frequency*, but there may be some timing offset between both bumps. For example, bump nr. 1 in Fig. 3(a) ($t = 10.7$s) should be paired with bump nr. 3 ($t = 10.9$s) and not with nr. 2 ($t = 10.8$s), since the former is much closer in frequency than the latter. As a consequence, we a priori expect smaller values for $s_f$ than for $s_t$. We encode this prior information by means of conjugate priors for $s_t$ and $s_f$, i.e., scaled inverse chi-square distributions.

A factor graph of model (14) is shown in Fig. 4 (each edge represents a variable, each node corresponds to a factor of (14), as indicated by the arrows at the right hand side; we refer to [7] for an introduction to factor graphs). We omitted the edges for the (observed) variables $X_k$, $X_{k'}'$, $F_k$, $F_{k'}'$, $\Delta X_k$, $\Delta X_{k'}'$, $\Delta F_k$, and $\Delta F_{k'}'$ in order not to clutter the figure.

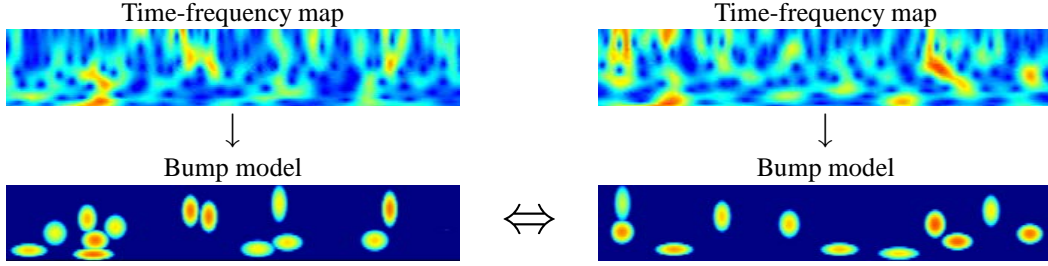

Figure 2: Two-dimensional stochastic event synchrony.

We determine the alignment $C = (C_{11}, C_{12}, \ldots, C_{nn'})$ and the parameters $\theta = (\delta_t, s_t, \delta_f, s_f)$ by maximum a posteriori (MAP) estimation:

$$(\hat{c}, \hat{\theta}) = \underset{c, \theta}{\arg\max}\, p(y, y', c, \theta), \tag{13}$$

where $p(y, y', c, \theta)$ is obtained from (14) by marginalizing over $b$ and $b'$:

$$p(y, y', c, \theta) \propto \prod_{k=1}^{n} \left(\beta\delta\left[\sum_{k'=1}^{n'} c_{kk'}\right] + \delta\left[\sum_{k'=1}^{n'} c_{kk'} - 1\right]\right) \prod_{k'=1}^{n'} \left(\beta\delta\left[\sum_{k=1}^{n} c_{kk'}\right] + \delta\left[\sum_{k=1}^{n} c_{kk'} - 1\right]\right)$$

$$\cdot \prod_{k=1}^{n} \prod_{k'=1}^{n'} \left( \mathcal{N}\left(\frac{x_{k'}' - x_k}{\Delta x_k + \Delta x_{k'}'}; \delta_t, s_t\right) \mathcal{N}\left(\frac{f_{k'}' - f_k}{\Delta f_k + \Delta f_{k'}'}; \delta_f, s_f\right) \right)^{c_{kk'}} p(\delta_t)p(s_t)p(\delta_f)p(s_f). \tag{14}$$

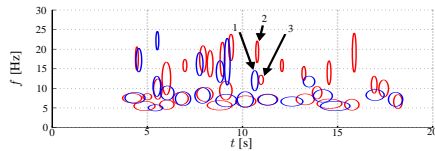

(a) Bump models of two EEG channels.

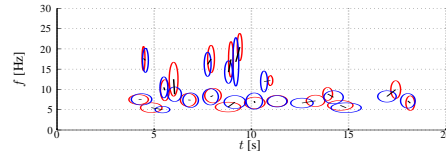

(b) Non-spurious bumps ($\rho_{\text{spur}} = 27\%$); the black lines connect the centers of non-spurious bumps.

Figure 3: Spurious and non-spurious activity.

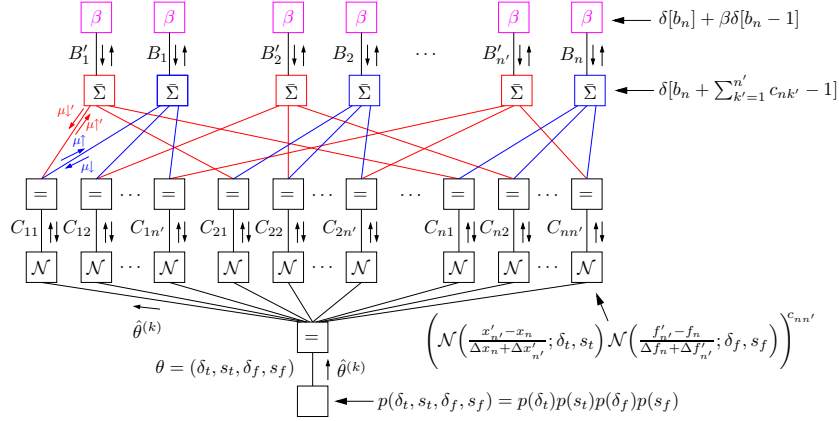

Figure 4: Factor graph of model (14).

From $\hat{c}$, we obtain the estimate $\hat{\rho}_{\text{spur}}$ as:

$$\hat{\rho}_{\text{spur}} = \frac{\sum_{k=1}^{n} \hat{b}_k + \sum_{k=1}^{n'} \hat{b}'_{k'}}{n + n'} = \frac{n + n' - 2\sum_{k=1}^{n}\sum_{k'=1}^{n'} \hat{c}_{kk'}}{n + n'}. \tag{15}$$

The MAP estimate (13) is intractable, and we try to obtain (13) by cyclic maximization: first, the parameters $\theta$ are initialized: $\hat{\delta}_t^{(0)} = 0 = \delta_f^{(0)}$, $\hat{s}_t^{(0)} = \hat{s}_{0,t}$, and $\hat{s}_f^{(0)} = s_{0,f}$, then one alternates the following two update rules until convergence (or until the available time has elapsed):

$$\hat{c}^{(i+1)} = \underset{c}{\operatorname{argmax}}\, p(y, y', c, \hat{\theta}^{(i)}) \tag{16}$$

$$\hat{\theta}^{(i+1)} = \underset{\theta}{\operatorname{argmax}}\, p(y, y', \hat{c}^{(i+1)}, \theta). \tag{17}$$

The estimate $\hat{\theta}^{(i+1)}$ (17) is available in closed-form; indeed, it is easily verified that the point estimates $\hat{\delta}_t^{(i+1)}$ and $\hat{\delta}_f^{(i+1)}$ are the (sample) mean of the timing and frequency offset respectively, computed over all pairs of non-spurious events. The estimates $\hat{s}_t^{(i+1)}$ and $\hat{s}_f^{(i+1)}$ are obtained similarly.

Update (16), i.e., finding the optimal pairwise alignment $C$ for *given* values $\hat{\theta}^{(i)}$ of the parameters $\theta$, is less straightforward: it involves an intractable combinatorial optimization problem. We attempt to solve that problem by applying the max-product algorithm to the (cyclic) factor graph depicted in Fig. 4 [7]. Let us first point out that, since the alignment $C$ is computed for given $\theta = \hat{\theta}^{(i)}$, the (upward) messages along the edges $\theta$ are the point estimate $\hat{\theta}^{(i)}$ (cf. (16)); equivalently, for the purpose of computing (16), one may remove the $\theta$ edges and the two bottom nodes in Fig. 4; the $\mathcal{N}$-nodes then become leaf nodes. The other messages in the graph are iteratively updated according to the generic max-product update rule [7].

The resulting inference algorithm for computing (16) is summarized in Table 1. The messages $\mu\uparrow(c_{kk'})$ and $\mu\uparrow'(c_{kk'})$ propagate *upward* along the edges $c_{kk'}$ towards the $\bar{\Sigma}$-nodes connected to the edges $B_k$ and $B'_{k'}$ respectively (see Fig. 4, left hand side); the messages $\mu\downarrow(c_{kk'})$ and $\mu\downarrow'(c_{kk'})$ propagate *downward* along the edges $c_{kk'}$ from the $\bar{\Sigma}$-nodes connected to the edges $B_k$ and $B'_{k'}$ respectively. After initialization (18) of the messages $\mu\uparrow(c_{kk'})$ and $\mu\uparrow'(c_{kk'})$ ($k = 1, 2, \ldots, n$; $k' = 1, 2, \ldots, n'$), one alternatively updates (i) the messages $\mu\downarrow(c_{kk'})$ (19) and $\mu\downarrow'(c_{kk'})$ (20), (ii) the messages $\mu\uparrow(c_{kk'})$ (21) and $\mu\uparrow'(c_{kk'})$ (22), until convergence; it is noteworthy that, although the max-product algorithm is not guaranteed to converge on cyclic graphs, we observed in our experiments (see Section 4) that alternating the updates (19)–(22) always converged to a fixed point. At last, one computes the marginals $p(c_{kk'})$ (23), and from the latter, one may determine the decisions $\hat{c}_{kk'}$ by greedy decimation.

## 4 Diagnosis of MCI from EEG

We analyzed rest eyes-closed EEG data recorded from 21 sites on the scalp based on the 10–20 system. The sampling frequency was 200 Hz, and the signals were bandpass filtered between 4

**Initialization**

$$\mu\!\uparrow\!(c_{kk'}) = \mu\!\uparrow'\!(c_{kk'}) \propto \left( \mathcal{N}\!\left(\frac{x'_{k'} - x_k}{\Delta x_k + \Delta x'_{k'}}; \delta_t, s_t\right) \mathcal{N}\!\left(\frac{f'_{k'} - f_k}{\Delta f_k + \Delta f'_{k'}}; \delta_f, s_f\right) \right)^{c_{kk'}} \quad (18)$$

**Iteratively compute messages until convergence**

A. Downward messages:

$$\begin{pmatrix} \mu\!\downarrow\!(c_{kk'} = 0) \\ \mu\!\downarrow\!(c_{kk'} = 1) \end{pmatrix} \propto \begin{pmatrix} \max\left(\beta, \max_{\ell' \neq k'} \mu\!\uparrow\!(c_{k\ell'} = 1)/\mu\!\uparrow\!(c_{k\ell'} = 0)\right) \\ 1 \end{pmatrix} \quad (19)$$

$$\begin{pmatrix} \mu\!\downarrow'\!(c_{kk'} = 0) \\ \mu\!\downarrow'\!(c_{kk'} = 1) \end{pmatrix} \propto \begin{pmatrix} \max\left(\beta, \max_{\ell \neq k} \mu\!\uparrow'\!(c_{\ell k'} = 1)/\mu\!\uparrow'\!(c_{\ell k'} = 0)\right) \\ 1 \end{pmatrix} \quad (20)$$

B. Upward messages:

$$\mu\!\uparrow\!(c_{kk'}) \propto \mu\!\downarrow'\!(c_{kk'}) \left( \mathcal{N}\!\left(\frac{x'_{k'} - x_k}{\Delta x_k + \Delta x'_{k'}}; \delta_t, s_t\right) \mathcal{N}\!\left(\frac{f'_{k'} - f_k}{\Delta f_k + \Delta f'_{k'}}; \delta_f, s_f\right) \right)^{c_{kk'}} \quad (21)$$

$$\mu\!\uparrow'\!(c_{kk'}) \propto \mu\!\downarrow\!(c_{kk'}) \left( \mathcal{N}\!\left(\frac{x'_{k'} - x_k}{\Delta x_k + \Delta x'_{k'}}; \delta_t, s_t\right) \mathcal{N}\!\left(\frac{f'_{k'} - f_k}{\Delta f_k + \Delta f'_{k'}}; \delta_f, s_f\right) \right)^{c_{kk'}} \quad (22)$$

**Marginals**

$$p(c_{kk'}) \propto \mu\!\downarrow\!(c_{kk'})\mu\!\downarrow'\!(c_{kk'}) \left( \mathcal{N}\!\left(\frac{x'_{k'} - x_k}{\Delta x_k + \Delta x'_{k'}}; \delta_t, s_t\right) \mathcal{N}\!\left(\frac{f'_{k'} - f_k}{\Delta f_k + \Delta f'_{k'}}; \delta_f, s_f\right) \right)^{c_{kk'}} \quad (23)$$

Table 1: Inference algorithm.

and 30Hz. The subjects comprised two study groups: the first consisted of a group of 22 patients diagnosed as suffering from MCI, who subsequently developed mild AD. The other group was a control set of 38 age-matched, healthy subjects who had no memory or other cognitive impairments. Pre-selection was conducted to ensure that the data were of a high quality, as determined by the presence of at least 20s of artifact free data. We computed a large variety of synchrony measures for both data sets; the results are summarized in Table 2. We report results for global synchrony, obtained by averaging the synchrony measures over 5 brain regions (frontal, temporal left and right, central, occipital). For SES, the bump models were clustered by means of the aggregation algorithm described in [17].

The strongest observed effect is a significantly higher degree of background noise ($\rho_{\text{spur}}$) in MCI patients, more specifically, a high number of spurious, non-synchronous oscillatory events (p = 0.00021). We verified that the SES measures are not correlated (Pearson $r$) with other synchrony measures ($p > 0.10$); in contrast to the other measures, SES quantifies the synchrony of oscillatory events (instead of more conventional amplitude or phase synchrony). Combining $\rho_{\text{spur}}$ with ffDTF yields good classification of MCI vs. Control patients (see Fig.5(a)). Interestingly, we did not observe a significant effect on the timing jitter $s_t$ of the non-spurious events (p = 0.91). In other words, AD seems to be associated with a significant increase of spurious background activity, while the *non-spurious* activity remains well synchronized. Moreover, only the non-spurious activity slows down (p = 0.0012; see Fig.5(c)), the average frequency of the spurious activity is not affected in MCI patients (see Fig.5(c)). In future work, we will verify those observations by means of additional data sets.

| Measure | Cross-correlation | Coherence | Phase Coherence | Corr-entropy | Wave-entropy | |
|---|---|---|---|---|---|---|
| p-value | **0.028*** | 0.060 | 0.72 | 0.27 | **0.012*** | |
| References | | | [16] | | [18] | [20] |
| Measure | Granger coherence | Partial Coherence | PDC | DTF | ffDTF | dDTF |
| p-value | 0.15 | 0.16 | 0.60 | 0.34 | **0.0012*** | **0.030*** |
| References | | | | [13] | | |
| Measure | Kullback-Leibler | Rényi | Jensen-Shannon | Jensen-Rényi | $I_W$ | $I$ |
| p-value | 0.072 | 0.076 | 0.084 | 0.12 | 0.060 | 0.080 |
| References | | | [23] | | | [22] |
| Measure | $N^k$ | $S^k$ | $H^k$ | S-estimator | | |
| p-value | **0.032*** | 0.29 | 0.090 | 0.33 | | |
| References | | [15] | | [21] | | |
| Measure | Hilbert Phase | Wavelet Phase | Evolution Map | Instantaneous Period | | |
| p-value | 0.15 | 0.082 | 0.072 | **0.020*** | | |
| References | | [24] | | [19] | | |
| Measure | $s_t$ | $\rho_{\text{spur}}$ | | | | |
| p-value | 0.91 | **0.00021*** | | | | |

Table 2: Sensitivity of synchrony measures for early prediction of AD (p-values for Mann-Whitney test; * and ** indicate $p < 0.05$ and $p < 0.005$ respectively). $N^k$, $S^k$, and $H^k$ are three measures of nonlinear interdependence [15].

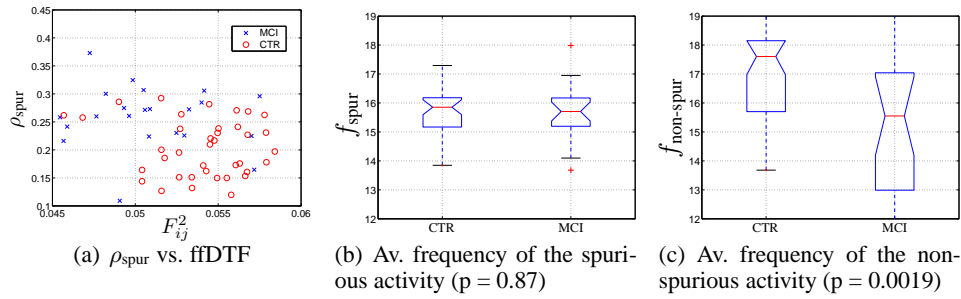

(a) $\rho_{\text{spur}}$ vs. ffDTF

(b) Av. frequency of the spurious activity (p = 0.87)

(c) Av. frequency of the non-spurious activity (p = 0.0019)

Figure 5: Results.

# References

[1] F. Varela, J. P. Lachaux, E. Rodriguez, and J. Martinerie, "The Brainweb: Phase Synchronization and Large-Scale Integration", *Nature Reviews Neuroscience,* 2(4):229–39, 2001.

[2] W. Singer, "Consciousness and the Binding Problem," *Annals of the New York Academy of Sciences,* 929:123–146, April 2001.

[3] M. Abeles, H. Bergman, E. Margalit, and E. Vaadia, "Spatiotemporal Firing Patterns in the Frontal Cortex of Behaving Monkeys," *J. Neurophysiol,* 70(4):1629–1638. 1993.

[4] S. Amari, H. Nakahara, S. Wu, and Y. Sakai, "Synchronous Firing and Higher-Order Interactions in Neuron Pool," *Neural Computation,* 15:127–142, 2003.

[5] H. Matsuda, "Cerebral Blood Flow and Metabolic Abnormalities in Alzheimer's Disease," *Ann. Nucl. Med.,* vol. 15, pp. 85–92, 2001.

[6] J. Jong, "EEG Dynamics in Patients with Alzheimer's Disease," *Clinical Neurophysiology,* 115:1490–1505 (2004).

[7] H.-A. Loeliger, "An Introduction to Factor Graphs," *IEEE Signal Processing Magazine*, Jan. 2004, pp. 28–41.

[8] C. S. Myers and L. R. Rabiner, "A Comparative Study of Several Dynamic Time-Warping Algorithms for Connected Word Recognition," *The Bell System Technical Journal,* 60(7):1389–1409, September 1981.

[9] J. D. Victor and K. P. Purpura, "Metric-space Analysis of Spike Trains: Theory, Algorithms, and Application," *Network: Comput. Neural Systems,* 8:17, 164, 1997.

[10] H. P. C. Robinson, "The Biophysical Basis of Firing Variability in Cortical Neurons," Chapter 6 in *Computational Neuroscience: A Comprehensive Approach,* Mathematical Biology & Medicine Series, Edited By Jianfeng Feng, Chapman & Hall/CRC, 2003.

[11] E. Pereda, R. Q. Quiroga, and J. Bhattacharya, "Nonlinear Multivariate Analysis of Neurophsyiological Signals," *Progress in Neurobiology,* 77 (2005) 1–37.

[12] M. Breakspear, "Dynamic Connectivity in Neural Systems: Theoretical and Empirical Considerations," *Neuroinformatics,* vol. 2, no. 2, 2004.

[13] M. Kamiński and Hualou Liang, "Causal Influence: Advances in Neurosignal Analysis," *Critical Review in Biomedical Engineering,* 33(4):347–430 (2005).

[14] C. J. Stam, "Nonlinear Dynamical Analysis of EEG and MEG: Review of an Emerging Field," *Clinical Neurophysiology* 116:2266–2301 (2005).

[15] R. Q. Quiroga, A. Kraskov, T. Kreuz, and P. Grassberger, "Performance of Different Synchronization Measures in Real Data: A Case Study on EEG Signals," *Physical Review E,* vol. 65, 2002.

[16] P. Nunez and R. Srinivasan, *Electric Fields of the Brain: The Neurophysics of EEG,* Oxford University Press, 2006.

[17] F. Vialatte, C. Martin, R. Dubois, J. Haddad, B. Quenet, R. Gervais, and G. Dreyfus, "A Machine Learning Approach to the Analysis of Time-Frequency Maps, and Its Application to Neural Dynamics," *Neural Networks,* 2007, 20:194–209.

[18] Jian-Wu Xu, H. Bakardjian, A. Cichocki, and J. C. Principe, "EEG Synchronization Measure: a Reproducing Kernel Hilbert Space Approach," submitted to *IEEE Transactions on Biomedical Engineering Letters*, Sept. 2006.

[19] M. G. Rosenblum, L. Cimponeriu, A. Bezerianos, A. Patzak, and R. Mrowka, "Identification of Coupling Direction: Application to Cardiorespiratory Interaction," *Physical Review E,* 65 041909, 2002.

[20] C. S. Herrmann, M. Grigutsch, and N. A. Busch, "EEG Oscillations and Wavelet Analysis," in Todd Handy (ed.) *Event-Related Potentials: a Methods Handbook,* pp. 229-259, Cambridge, MIT Press, 2005.

[21] C. Carmeli, M. G. Knyazeva, G. M. Innocenti, and O. De Feo, "Assessment of EEG Synchronization Based on State-Space Analysis," *Neuroimage,* 25:339–354 (2005).

[22] A. Kraskov, H. Stögbauer, and P. Grassberger, "Estimating Mutual Information," *Phys. Rev. E* 69 (6) 066138, 2004.

[23] S. Aviyente, "A Measure of Mutual Information on the Time-Frequency Plane," *Proc. of ICASSP 2005,* vol. 4, pp. 481–484, March 18–23, 2005, Philadelphia, PA, USA.

[24] J.-P. Lachaux, E. Rodriguez, J. Martinerie, and F. J. Varela, "Measuring Phase Synchrony in Brain Signals," *Human Brain Mapping* 8:194208 (1999).

